# Joint Probabilistic Curve Clustering and Alignment

**Scott Gaffney and Padhraic Smyth**
School of Information and Computer Science
University of California, Irvine, CA 92697-3425
{sgaffney,smyth}@ics.uci.edu

## Abstract

Clustering and prediction of sets of curves is an important problem in many areas of science and engineering. It is often the case that curves tend to be misaligned from each other in a continuous manner, either in space (across the measurements) or in time. We develop a probabilistic framework that allows for joint clustering and continuous alignment of sets of curves in curve space (as opposed to a fixed-dimensional feature-vector space). The proposed methodology integrates new probabilistic alignment models with model-based curve clustering algorithms. The probabilistic approach allows for the derivation of consistent EM learning algorithms for the joint clustering-alignment problem. Experimental results are shown for alignment of human growth data, and joint clustering and alignment of gene expression time-course data.

## 1 Introduction

We introduce a novel methodology for the clustering and prediction of sets of smoothly varying curves while jointly allowing for the learning of sets of continuous curve transformations. Our approach is to formulate models for both the clustering and alignment sub-problems and integrate them into a unified probabilistic framework that allows for the derivation of consistent learning algorithms. The alignment sub-problem is handled with the introduction of a novel curve alignment procedure employing model priors over the set of possible alignments leading to the derivation of EM learning algorithms that formalize the so-called *Procrustes* approach for curve data [1]. These alignment models are then integrated into a finite mixture model setting in which the clustering is carried out. We make use of both polynomial and spline regression mixture models to complete the joint clustering-alignment framework.

The following simple illustrative example demonstrates the importance of jointly handling the clustering-alignment problem as opposed to treating alignment and clustering separately. Figure 1(a) shows a simulated set of curves which have been subjected to random translations in time. The underlying generative model contains three clusters each described by a cubic polynomial (not shown). Figure 1(b) shows the output of the proposed *joint* EM algorithm introduced in this paper, where curves have been simultaneously aligned and clustered. The algorithm recovers the hidden labels and alignments near-perfectly in this case. On the other hand, Figure 1(c) shows the result of first clustering

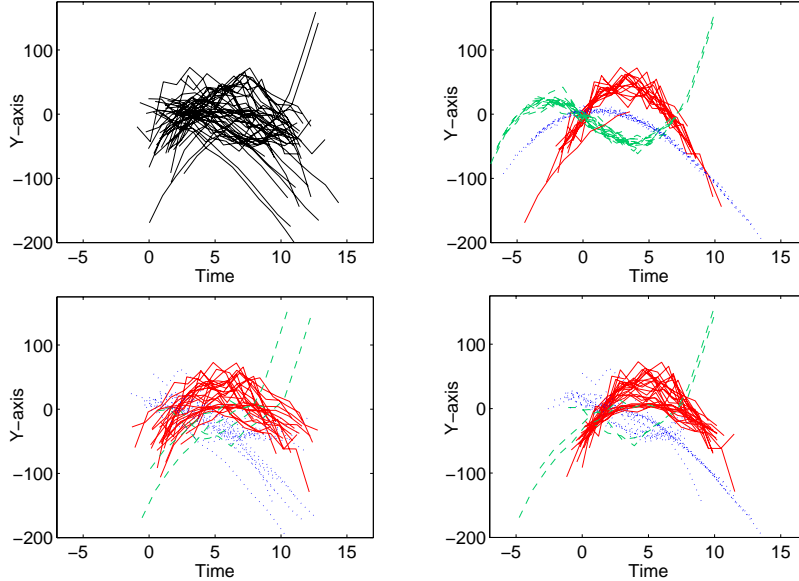

Figure 1: Comparison of joint EM and sequential clustering-alignment: (a, top-left) unlabelled simulated data with hidden alignments; (b, top-right) solution recovered by joint EM; (c, bottom-left) partial solution after clustering first, and (d, bottom-right) final solution after aligning clustered data in (c).

the unaligned data in Figure 1(b), while Figure 1(d) shows the final result of aligning each of the found clusters individually. The sequential approach results in significant misclassification and incorrect alignment demonstrating that a two-stage approach can be quite suboptimal when compared to a joint clustering-alignment methodology. (Similar results, not shown, are obtained when the curves are first aligned and then clustered—see [2] for full details.)

There has been little prior work on the specific problem of joint curve clustering and alignment, but there is related work in other areas. For example, clustering of gene-expression time profiles with mixtures of splines was addressed in [3]. However, alignment was only considered as a post-processing step to compare cluster results among related datasets. In image analysis, the *transformed mixture of Gaussians* (TMG) model uses a probabilistic framework and an EM algorithm to jointly learn clustering and alignment of image patches subject to various forms of linear transformations [4]. However, this model only considers sets of transformations in discrete pixel space, whereas we are focused on curve modelling that allows for arbitrary *continuous* alignment in time and space. Another branch of work in image analysis focuses on the problem of estimating correspondences of points across images [5] (or vertices across graphs [6]), using EM or deterministic annealing algorithms. The results we describe here differ primarily in that (a) we focus specifically on sets of curves rather than image data (generally making the problem more tractable), (b) we focus on clustering and alignment rather than just alignment, (c) we allow continuous affine transformations in time and measurement space, and (d) we have a fully generative probabilistic framework allowing for (for example) the incorporation of informative priors on transformations if such prior information exists.

In earlier related work we developed general techniques for curve clustering (e.g., [7]) and also proposed techniques for transformation-invariant curve clustering with discrete time alignment and Gaussian mixture models for curves [8, 9]. In this paper we provide

a much more general framework that allows for continuous alignment in both time and measurement space for a general class of "cluster shape" models, including polynomials and splines.

## 2 Joint clustering and alignment

It is useful to represent curves as variable-length vectors. In this case, $\mathbf{y}_i$ is a curve that consists of a sequence of $n_i$ observations or measurements. The $j$-th measurement of $\mathbf{y}_i$ is denoted by $y_{ij}$ and is usually taken to be univariate (the generalization to multivariate observations is straightforward). The associated covariate of $\mathbf{y}_i$ is written as $\mathbf{x}_i$ in the same manner. $\mathbf{x}_i$ is often thought of as time so that $x_{ij}$ gives the time at which $y_{ij}$ was observed.

Regression mixture models can be effectively used to cluster this type of curve data [10]. In the standard setup, $\mathbf{y}_i$ is modelled using a normal (Gaussian) regression model in which $\mathbf{y}_i = \mathbf{X}_i\boldsymbol{\beta} + \boldsymbol{\epsilon}_i$, where $\boldsymbol{\beta}$ is a $(p+1) \times 1$ coefficient vector, $\boldsymbol{\epsilon}_i$ is a zero-mean Gaussian noise variable, and $\mathbf{X}_i$ is the regression matrix. The form of $\mathbf{X}_i$ depends on the type of regression model employed. For polynomial regression, $\mathbf{X}_i$ is often associated with the standard Vandermonde matrix; and for spline regression, $\mathbf{X}_i$ takes the form of a spline-basis matrix (see, e.g., [7] for more details). The mixture model is completed by repeating this model over $K$ clusters and indexing the parameters by $k$ so that, for example, $\mathbf{y}_i = \mathbf{X}_i\boldsymbol{\beta}_k + \boldsymbol{\epsilon}_i$ gives the regression model for $\mathbf{y}_i$ under the $k$-th cluster.

B-splines [11] are particularly efficient for computational purposes due to the block-diagonal basis matrices that result. Using B-splines, the curve point $y_{ij}$ can be represented as the linear combination $y_{ij} = \mathbf{B}'_{ij}\mathbf{c}$, in which the vector $\mathbf{B}_{ij}$ gives the vector of B-spline basis functions evaluated at $x_{ij}$, and $\mathbf{c}$ gives the spline coefficient vector [2]. The full curve $\mathbf{y}_i$ can then be written compactly as $\mathbf{y}_i = \mathbf{B}_i\mathbf{c}$ in which the spline basis matrix takes the form $\mathbf{B}_i = [\mathbf{B}_{i1} \cdots \mathbf{B}_{in_i}]'$. Spline regression models can be easily integrated into the regression mixture model framework by equating the regression matrix $\mathbf{X}_i$ with the spline basis matrix $\mathbf{B}_i$. In what follows, we use the more general notation $\mathbf{X}_i$ in favor of the more specific $\mathbf{B}_i$.

### 2.1 Joint model definition

The joint clustering-alignment model definition is based on a regression mixture model that has been augmented with up to four individual random transformation parameters or variables $(a_i, b_i, c_i, d_i)$. The $a_i$ and $b_i$ allow for scaling and translation in time, while the $c_i$ and $d_i$ allow for scaling and translation in measurement space. The model definition takes the form

$$\mathbf{y}_i = c_i[\![a_i\mathbf{x}_i - b_i]\!]\boldsymbol{\beta}_k + d_i + \boldsymbol{\epsilon}_i, \tag{1}$$

in which $[\![a_i\mathbf{x}_i - b_i]\!]$ represents the regression matrix $\mathbf{X}_i$ (either spline or polynomial) evaluated at the transformed time $a_i\mathbf{x}_i - b_i$. Below we use the matrix $\mathcal{X}_i$ to denote $[\![a_i\mathbf{x}_i - b_i]\!]$ when parsimony is required. It is assumed that $\boldsymbol{\epsilon}_i$ is a zero-mean Gaussian vector with covariance $\sigma_k^2\mathbf{I}$.

The conditional density

$$p_k(\mathbf{y}_i|a_i, b_i, c_i, d_i) = \mathcal{N}(\mathbf{y}_i|c_i[\![a_i\mathbf{x}_i - b_i]\!]\boldsymbol{\beta}_k + d_i, \sigma_k^2\mathbf{I}) \tag{2}$$

gives the probability density of $\mathbf{y}_i$ when all the transformation parameters (as well as cluster membership) are known. (Note that the density on the left is implicitly conditioned on an appropriate set of parameters—this is always assumed in what follows.) In general, the values for the transformation parameters are unknown. Treating this as a standard hidden-data problem, it is useful to think of each of the transformation parameters as random variables that are curve-specific but with "population-level" prior probability distributions. In this

way, the transformation parameters and the model parameters can be learned simultaneously in an efficient manner using EM.

## 2.2 Transformation priors

Priors are attached to each of the transformation variables in such a way that the identity transformation is the most likely transformation. A useful prior for this is the Gaussian density $\mathcal{N}(\mu, \sigma^2)$ with mean $\mu$ and variance $\sigma^2$. The time transformation priors are specified as

$$a_i \sim \mathcal{N}(1, r_k^2), \quad b_i \sim \mathcal{N}(0, s_k^2), \tag{3}$$

and the measurement space priors are given as

$$c_i \sim \mathcal{N}(1, u_k^2) \quad , d_i \sim \mathcal{N}(0, v_k^2). \tag{4}$$

Note that the identity transformation is indeed the most likely. All of the variance parameters are cluster-specific in general; however, any subset of these parameters can be "tied" across clusters if desired in a specific application. Note that these priors technically allow for negative scaling in time and in measurement space. In practice this is typically not a problem, though one can easily specify other priors (e.g., log-normal) to strictly disallow this possibility. It should be noted that each of the prior variance parameters are learned from the data in the ensuing EM algorithm. We do not make use of hyperpriors for these prior parameters; however, it is straightforward to extend the method to allow hyperpriors if desired.

## 2.3 Full probability model

The joint density of $\mathbf{y}_i$ and the set of transformation variables $\Phi_i = \{a_i, b_i, c_i, d_i\}$ can be written succinctly as

$$p_k(\mathbf{y}_i, \Phi_i) = p_k(\mathbf{y}_i|\Phi_i)p_k(\Phi_i), \tag{5}$$

where $p_k(\Phi_i) = \mathcal{N}(a_i|1, r_k^2)\mathcal{N}(b_i|0, s_k^2)\mathcal{N}(c_i|1, u_k^2)\mathcal{N}(d_i|0, v_k^2)$. The space transformation parameters can be integrated-out of (5) resulting in the marginal of $\mathbf{y}_i$ conditioned only on the time transformation parameters. This conditional marginal takes the form

$$
\begin{aligned}
p_k(\mathbf{y}_i|a_i, b_i) &= \int \int p_k(\mathbf{y}_i, c_i, d_i|a_i, b_i)\, dc_i, dd_i \\
&= \mathcal{N}(\mathbf{y}_i|\mathcal{X}_i\boldsymbol{\beta}_k, \mathbf{U}_{ik} + \mathbf{V}_k - \sigma_k^2\mathbf{I}),
\end{aligned}
\tag{6}
$$

with $\mathbf{U}_{ik} = u_k^2 \mathcal{X}_i \boldsymbol{\beta}_k \boldsymbol{\beta}_k' \mathcal{X}_i' + \sigma_k^2 \mathbf{I}$ and $\mathbf{V}_k = v_k^2 \mathbf{1}\mathbf{1}' + \sigma_k^2 \mathbf{I}$. The unconditional (though, still cluster-dependent) marginal for $\mathbf{y}_i$ cannot be computed analytically since $a_i, b_i$ cannot be analytically integrated-out. Instead, we use numerical Monte Carlo integration for this task. The resulting unconditional marginal for $\mathbf{y}_i$ can be approximated by

$$
\begin{aligned}
p_k(\mathbf{y}_i) &= \int \int p_k(\mathbf{y}_i|a_i, b_i)p_k(a_i)p_k(b_i)\, da_i\, db_i \\
&\approx \frac{1}{M}\sum_m p_k(\mathbf{y}_i|a_i^{(m)}, b_i^{(m)}),
\end{aligned}
\tag{7}
$$

where the $M$ Monte Carlo samples are taken according to

$$a_i^{(m)} \sim \mathcal{N}(1, r_k^2), \quad \text{and} \quad b_i^{(m)} \sim \mathcal{N}(0, s_k^2), \quad \text{for } m = 1, \dots, M. \tag{8}$$

A mixture results when cluster membership is unknown:

$$p(\mathbf{y}_i) = \sum_k \alpha_k p_k(\mathbf{y}_i). \tag{9}$$

The log-likelihood of all $n$ curves $Y = \{\mathbf{y}_i\}$ follows directly from this approximation and takes the form

$$\log p(Y) \approx \sum_i \log \sum_{mk} \alpha_k p_k(\mathbf{y}_i|a_i^{(m)}, b_i^{(m)}) - n\log M. \tag{10}$$

### 2.4 EM algorithm

We derive an EM algorithm that simultaneously allows the learning of both the model parameters and the transformation variables $\Phi$ with time-complexity that is linear in the total number of data points $N = \sum_i n_i$. First, let $z_i$ give the cluster membership for curve $\mathbf{y}_i$. Now, regard the transformation variables $\{\Phi_i\}$ as well as the cluster memberships $\{z_i\}$ as being hidden. The complete-data log-likelihood function is defined as the joint log-likelihood of $Y$ and the hidden data $\{\Phi_i, z_i\}$. This can be written as the sum over all $n$ curves of the log of the product of $\alpha_{z_i}$ and the cluster-dependent joint density in (5). This function takes the form

$$\mathcal{L}_c = \sum_i \log \alpha_{z_i} p_{z_i}(\mathbf{y}_i|\Phi_i) \, p_{z_i}(\Phi_i). \tag{11}$$

In the E-step, the posterior $p(\Phi_i, z_i|\mathbf{y}_i)$ is calculated and then used to take the posterior expectation of Equation (11). This expectation is then used in the M-step to calculate the re-estimation equations for updating the model parameters $\{\boldsymbol{\beta}_k, \sigma_k^2, r_k^2, s_k^2, u_k^2, v_k^2\}$.

### 2.5 E-step

The posterior $p(\Phi_i, z_i|\mathbf{y}_i)$ can be factorized as $p_{z_i}(\Phi|\mathbf{y}_i)p(z_i|\mathbf{y}_i)$. The second factor is the membership probability $w_{ik}$ that $\mathbf{y}_i$ was generated by cluster $k$. It can be rewritten as $p(z_i = k|\mathbf{y}_i) \propto p_k(\mathbf{y}_i)$ and evaluated using Equation (7). The first factor requires a bit more work. Further factoring reveals that $p_{z_i}(\Phi|\mathbf{y}_i) = p_{z_i}(c_i, d_i|a_i, b_i, \mathbf{y}_i)p_{z_i}(a_i, b_i|\mathbf{y}_i)$. The new first factor $p_{z_i}(c_i, d_i|a_i, b_i, \mathbf{y}_i)$ can be solved for exactly by noting that it is proportional to a bivariate normal distribution for each $z_i$ [2]. The new second factor $p_{z_i}(a_i, b_i|\mathbf{y}_i)$ cannot, in general, be solved for analytically, so instead we use an approximation.

The fact that posterior densities tend towards highly peaked Gaussian densities has been widely noted (e.g, [12]) and leads to the normal approximation of posterior densities. To make the approximation here, the vector $(\hat{a}_{ik}, \hat{b}_{ik})$ representing the multi-dimensional mode of $p_k(a_i, b_i|\mathbf{y}_i)$, the covariance matrix $V_{a_i b_i}^{(k)}$ for $(\hat{a}_{ik}, \hat{b}_{ik})$, and the separate variances $V_{a_{ik}}, V_{b_{ik}}$ must be found. These can readily be estimated using a Nelder-Mead optimization method. Experiments have shown this approximation works well across a variety of experimental and real-world data sets [2].

The above calculations of the posterior $p(\Phi_i, z_i|\mathbf{y}_i)$ allow the posterior expectation of the complete-data log-likelihood in Equation (11) to be solved for. This expectation results in the so-called $Q$-function which is maximized in the M-step. Although the derivation is quite complex, the $Q$-function can be calculated exactly for polynomial regression [2]; for spline regression, the basis functions do not afford an exact formula for the solution of the $Q$-function. However, in the spline case, removal of a few problematic variance terms gives an efficient approximation (the interested reader is referred to [2] for more details).

### 2.6 M-step

The M-step is straightforward since most of the hard work is done in the E-step. The $Q$-function is maximized over the set of parameters $\{\boldsymbol{\beta}_k, \sigma_k^2, r_k^2, s_k^2, u_k^2, v_k^2\}$ for $1 \leq k \leq K$. The derived solutions are as follows:

$$\hat{r}_k^2 = \frac{1}{\sum_i w_{ik}} \sum_i w_{ik} \left[\hat{a}_{ik}^2 + V_{a_{ik}}\right], \quad \hat{s}_k^2 = \frac{1}{\sum_i w_{ik}} \sum_i w_{ik} \left[\hat{b}_{ik}^2 + V_{b_{ik}}\right],$$

$$\hat{u}_k^2 = \frac{1}{\sum_i w_{ik}} \sum_i w_{ik} \left[\hat{c}_{ik}^2 + V_{c_{ik}}\right], \quad \hat{v}_k^2 = \frac{1}{\sum_i w_{ik}} \sum_i w_{ik} \left[\hat{d}_{ik}^2 + V_{d_{ik}}\right],$$

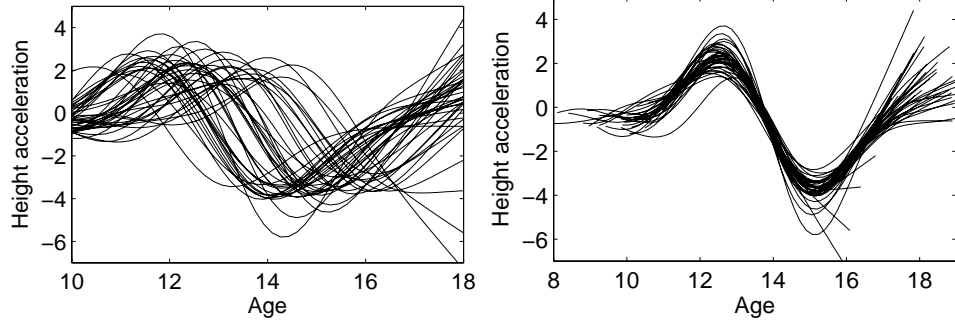

Figure 2: Curves measuring the height acceleration for 39 boys; (left) smoothed versions of raw observations, (right) automatically aligned curves.

$$\hat{\boldsymbol{\beta}}_k = \left[\sum_i w_{ik}\hat{c}_{ik}^2\hat{\mathcal{X}}_{ik}'\hat{\mathcal{X}}_{ik} + \mathbf{V_{xx}}_i\right]^{-1}\left[\sum_i w_{ik}\hat{c}_{ik}\hat{\mathcal{X}}_{ik}'(\mathbf{y}_i - \hat{d}_{ik}) + \mathbf{V}_{\mathbf{x}i}'\mathbf{y}_i - \mathbf{V}_{\mathbf{x}cd}'\mathbf{1}\right],$$

and

$$\hat{\sigma}_k^2 = \frac{1}{\sum_i w_{ik}n_i}\sum_i w_{ik}\left[\left\|\mathbf{y}_i - \hat{c}_{ik}\hat{\mathcal{X}}_{ik}\boldsymbol{\beta} - \hat{d}_{ik}\right\|^2\right.$$
$$\left. - 2\mathbf{y}_i'\mathbf{V}_{\mathbf{x}i}\hat{\boldsymbol{\beta}}_k + \hat{\boldsymbol{\beta}}_k'\mathbf{V_{xx}}_i\hat{\boldsymbol{\beta}}_k + 2\hat{\boldsymbol{\beta}}_k'\mathbf{V}_{\mathbf{x}cd}\mathbf{1} + n_iV_{d_{ik}}\right],$$

where $\hat{\mathcal{X}}_{ik} = [\![\hat{a}_{ik}\mathbf{x}_i - \hat{b}_{ik}]\!]$, and $\mathbf{V_{xx}}_i, \mathbf{V}_{\mathbf{x}i}, \mathbf{V}_{\mathbf{x}cd}$ are special "variance" matrices whose components are functions of the posterior expectations of $\Phi$ calculated in the E-step (the exact forms of these matrices can be found in [2]).

## 3   Experimental results and conclusions

The results of a simple demonstration of EM-based alignment (using splines and the learning algorithm of the previous section, but with no clustering) are shown in Figure 2. In the left plot are a set of smoothed curves representing the acceleration of height for each of 39 boys whose heights were measured at 29 observation times over the ages of 1 to 18 [1]. Notice that the curves share a similar shape but seem to be misaligned in time due to individual growth dynamics. The right plot shows the same acceleration curves after processing from our spline alignment model using quartic splines with 8 uniformly spaced knots allowing for a maximum time translation of 2 units. The $x$-axis in this plot can be seen as canonical (or "average") age. The aligned curves in the right plot of Figure 2 represent the average behavior in a much clearer way. For example, it appears there is an interval of 2.5 years from peak (age 12.5) to trough (age 15) that describes the average cycle that all boys go through. The results demonstrate that it is common for important features of curves to be randomly translated in time and that it is possible to use the data to recover these underlying hidden transformations using our alignment models.

Next we briefly present an application of the joint clustering-alignment model to the problem of gene expression clustering. We analyze the alpha arrest data described in [13] that captures gene expression levels at 7 minute intervals for two consecutive cell cycles (totaling 17 measurements per gene). Clustering is often used in gene expression analysis to reveal groups of genes with similar profiles that may be physically related to the same underlying biological process (e.g., [13]). It is well-known that time-delays play an impor-

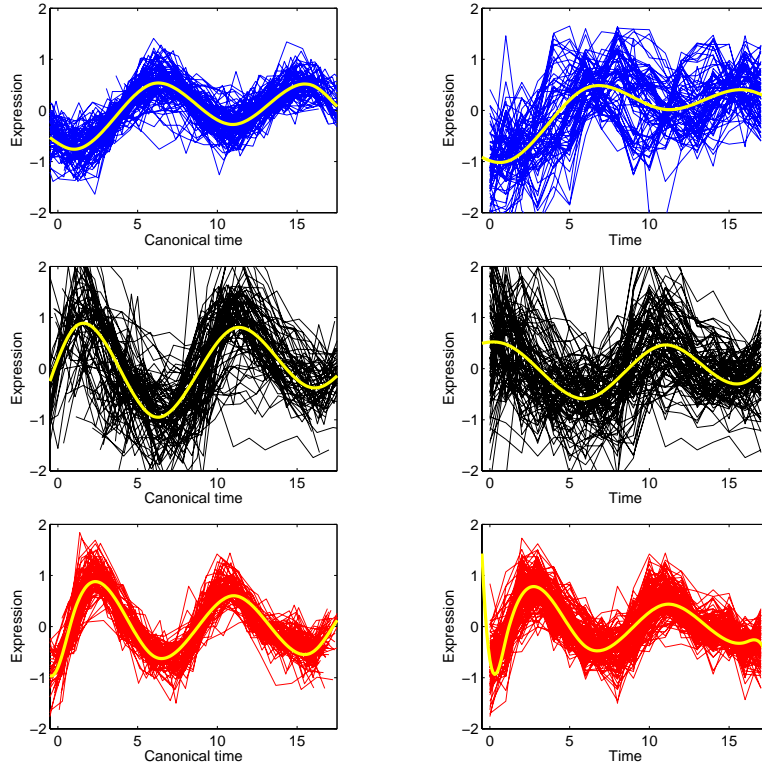

Figure 3: Three clusters for the time translation alignment model (left) and the non-alignment model (right).

tant role in gene regulation, and thus, curves measured over time which represent the same process may often be misaligned from each other. [14].

Since these gene expression data are already normalized, we did not allow for transformations in measurement space. We only allowed for translations in time since experts do not expect scaling in time to be a factor in these data. For the curve model, cubic splines with 6 uniformly spaced knots across the interval from $-4$ to 21 were chosen, allowing for a maximum time translation of 4 units. Due to limited space, we present a single case of comparison between a standard spline regression mixture model (SRM) and an SRM that jointly allows for time translations. Ten random starts of EM were allowed for each algorithm with the highest likelihood model selected for comparison for each algorithm. It is common to assume that there are five distinct clusters of genes in these data; as such we set $K = 5$ for each algorithm [13].

Three of the resulting clusters from the two methods are shown in Figure 3. The left column of the figure shows the output from the joint clustering-alignment model, while the right column shows the output from the standard cluster model. It is apparent that the time-aligned clusters represent the mean behavior more accurately. The overall cluster variance is much lower than in the non-aligned clustering. The results also demonstrate the appearance of cluster-dependent alignment effects. Out-of-sample experiments (not shown here) show that the joint model produces better predictive models than the standard clustering method. Experimental results on a variety of other data sets are provided in [2], including applications to clustering of cyclone trajectories.

# 4 Conclusions

We proposed a general probabilistic framework for joint clustering and alignment of sets of curves. The experimental results indicate that the approach provides a new and useful tool for curve analysis in the face of underlying hidden transformations. The resulting EM-based learning algorithms have time-complexity that is linear in the number of measurements—in contrast, many existing curve alignment algorithms themselves are $O(n^2)$ (e.g., dynamic time warping) without regard to clustering. The incorporation of splines gives the method an overall non-parametric freedom which leads to general applicability.

### Acknowledgements

This material is based upon work supported by the National Science Foundation under grants No. SCI-0225642 and IIS-0431085.

# References

[1] J.O. Ramsay and B. W. Silverman. *Functional Data Analysis*. Springer-Verlag, New York, NY, 1997.

[2] Scott J. Gaffney. *Probabilistic Curve-Aligned Clustering and Prediction with Regression Mixture Models*. Ph.D. Dissertation, University of California, Irvine, 2004.

[3] Z. Bar-Joseph et al. A new approach to analyzing gene expression time series data. *Journal of Computational Biology*, 10(3):341–356, 2003.

[4] B. J. Frey and N. Jojic. Transformation-invariant clustering using the EM algorithm. *IEEE Trans. PAMI*, 25(1):1–17, January 2003.

[5] H. Chui, J. Zhang, and A. Rangarajan. Unsupervised learning of an atlas from unlabeled point-sets. *IEEE Trans. PAMI*, 26(2):160–172, February 2004.

[6] A. D. J. Cross and E. R. Hancock. Graph matching with a dual-step EM algorithm. *IEEE Trans. PAMI*, 20(11):1236–1253, November 1998.

[7] S. J. Gaffney and P. Smyth. Curve clustering with random effects regression mixtures. In C. M. Bishop and B. J. Frey, editors, *Proc. Ninth Inter. Workshop on Artificial Intelligence and Stats*, Key West, FL, January 3–6 2003.

[8] D. Chudova, S. J. Gaffney, and P. J. Smyth. Probabilistic models for joint clustering and time-warping of multi-dimensional curves. In *Proc. of the Nineteenth Conference on Uncertainty in Artificial Intelligence (UAI-2003), Acapulco, Mexico, August 7–10*, 2003.

[9] D. Chudova, S. J. Gaffney, E. Mjolsness, and P. J. Smyth. Translation-invariant mixture models for curve clustering. In *Proc. Ninth ACM SIGKDD Inter. Conf. on Knowledge Discovery and Data Mining, Washington D.C., August 24–27*, New York, 2003. ACM Press.

[10] S. Gaffney and P. Smyth. Trajectory clustering with mixtures of regression models. In Surajit Chaudhuri and David Madigan, editors, *Proc. Fifth ACM SIGKDD Inter. Conf. on Knowledge Discovery and Data Mining, August 15–18*, pages 63–72, N.Y., 1999. ACM Press.

[11] P. H. C. Eilers. and B. D. Marx. Flexible smoothing with B-splines and penalties. *Statistical Science*, 11(2):89–121, 1996.

[12] A. Gelman, J. B. Carlin, H. S. Stern, and D. B. Rubin. *Bayesian Data Analysis*. Chapman & Hall, New York, NY, 1995.

[13] P. T. Spellman et al. Comprehensive identification of cell cycle-regulated genes of the yeast Saccharomyces cerevisiae by microarray hybridization. *Molec. Bio. Cell*, 9(12):3273–3297, December 1998.

[14] J. Aach and G. M. Church. Aligning gene expression time series with time warping algorithms. *Bioinformatics*, 17(6):495–508, 2001.
